# A Model for Learning the Semantics of Pictures

**V. Lavrenko, R. Manmatha, J. Jeon**
Center for Intelligent Information Retrieval
Computer Science Department,
University of Massachusetts Amherst
{lavrenko,manmatha,jeon}@cs.umass.edu

## Abstract

We propose an approach to learning the semantics of images which allows us to automatically annotate an image with keywords and to retrieve images based on text queries. We do this using a formalism that models the generation of annotated images. We assume that every image is divided into regions, each described by a continuous-valued feature vector. Given a training set of images with annotations, we compute a joint probabilistic model of image features and words which allow us to predict the probability of generating a word given the image regions. This may be used to automatically annotate and retrieve images given a word as a query. Experiments show that our model significantly outperforms the best of the previously reported results on the tasks of automatic image annotation and retrieval.

## 1  Introduction

Historically, librarians have retrieved images by first manually annotating them with keywords. Given a query, these annotations are used to retrieve appropriate pictures. Underlying this approach is the belief that the words associated (manually) with a picture essentially capture the semantics of the picture and any retrieval based on these keywords will, therefore, retrieve relevant pictures. Since manual image annotation is expensive, there has been great interest in coming up with automatic ways to retrieve images based on content. Queries based on image concepts like color or texture have been proposed for retrieving images by content but most users find it difficult to query using such visual attributes. Most people would prefer to pose text queries and find images relevant to those queries. For example, one should be able to pose a query like "find me cars on a race track". This is difficult if not impossible with many of the current image retrieval systems and hence has not led to widespread adoption of these systems. We propose a model which looks at the probability of associating words with image regions. Single pixels and regions are often hard to interpret. The surrounding context often simplifies the interpretation of regions as a specific objects. For example, the association of a region with the word tiger is increased by the fact that there is a grass region and a water region in the same image and should be decreased if instead there is a region corresponding to the interior of an aircraft. Thus the association of different regions provides context while the association of words with image regions provides meaning. Our model computes a joint probability of image features over different regions in an image using a training set and uses this joint probability to annotate and retrieve images.

More formally, we propose a statistical generative model to automatically learn the semantics of images - that is, for annotating and retrieving images based on a training set of images. We assume that an image is segmented into regions (although the regions could

simply be a partition of the image) and that features are computed over each of these regions. Given a training set of images with annotations, we show that probabilistic models allow us to predict the probability of generating a word given the features computed over different regions in an image. This may be used to automatically annotate and retrieve images given a word as a query. We show that the continuous relevance model - a statistical generative model related to relevance models in information retrieval - allows us to derive these probabilities in a natural way. The model proposed here directly associates continuous features with words and does not require an intermediate clustering stage. Experiments show that the annotation performance of this continuous relevance model is substantially better than any other model tested on the same data set. It is almost an order of magnitude better (in terms of mean precision) than a model based on word-blob co-occurrence model, more than two and a half times better than a state of the art model derived from machine translation and 1.6 times as good as a discrete version of the relevance model. The model also allows ranked retrieval in response to a text query and again performs much better than any other model in this regard. Our model permits us to automatically associate semantics (in terms of words) with pictures and is an important building step in performing automatic object recognition.

## 2 Related Work

Recently, there has been some work on automatically annotating images by looking at the probability of associating words with image regions. Mori *et al.* [9] proposed a *Co-occurrence Model* in which they looked at the co-occurrence of words with image regions created using a regular grid. Duygulu *et al* [4] proposed to describe images using a vocabulary of blobs. First, regions are created using a segmentation algorithm like normalized cuts. For each region, features are computed and then blobs are generated by clustering the image features for these regions across images. Each image is generated by using a certain number of these blobs. Their *Translation Model* applies one of the classical statistical machine translation models to translate from the set of keywords of an image to the set of blobs forming the image. Jeon et al [5] instead assumed that this could be viewed as analogous to the cross-lingual retrieval problem and used a cross-media relevance model (CMRM) to perform both image annotation and ranked retrieval. They showed that the performance of the model on the same dataset was considerably better than the models proposed by Duygulu *et al* [4] and Mori *et al.* [9]. Blei and Jordan [3] extended the Latent Dirichlet Allocation (LDA) Model and proposed a Correlation LDA model which relates words and images. This model assumes that a Dirichlet distribution can be used to generate a mixture of latent factors. This mixture of latent factors is then used to generate words and regions. EM is again used to estimate this model. Blei and Jordan show a few examples for labeling specific regions in an image.

The model proposed in this paper is called *Continuous-space Relevance Model* (CRM). The model is closely related to models proposed by [3, 5], but there are several important differences which we will highlight in the remainder of this section.

On the surface, CRM appears to be very similar to one of the intermediate models considered by Blei and Jordan [3]. Specifically, their *GM-mixture* model employs a nearly identical dependence structure among the random variables involved. However, the topological structure of CRM is quite different from the one employed by [3]. GM-mixture assumes a low-dimensional topology, leading to a fully-parametric model where 200 or so "latent aspects" are estimated using the EM algorithm. To contrast that, CRM makes no assumptions about the topological structure, and leads to a doubly non-parametric approach, where expectations are computed over every individual point in the training set. In that regard, CRM appears very similar to the cross-media relevance model (CMRM) [5], which is also doubly non-parametric. There are two significant differences between CRM and CMRM. First, CMRM is a discrete model and cannot take advantage of continuous features. In order to use CMRM for image annotation we have to quantize continuous feature

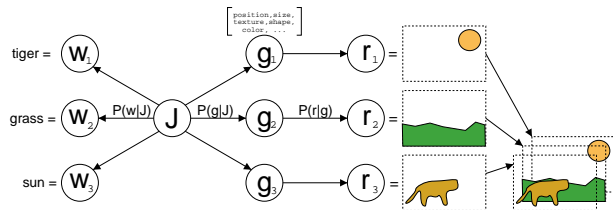

Figure 1: A generative model of annotated images. Words $w_j$ in the annotation are i.i.d. sampled from the underlying multinomial. Image pixels are produced by first picking a set of i.i.d. feature vectors $\{g_1 \ldots g_n\}$, then generating image regions $\{r_1 \ldots r_n\}$ from the feature vectors, and finally stacking the regions on top of each other.

vectors into a discrete vocabulary (similarly to the co-ocurrence and translation [4] models). CRM, on the other hand, directly models continuous features. The second difference is that CMRM relies on *clustering* of the feature vectors into *blobs*. Annotation quality of the CMRM is very sensitive to clustering errors, and depends on being able to a-priori select the right cluster granularity: too many clusters will result in exptreme sparseness of the space, while too few will lead us to confuse different objects in the images. CRM does not rely on clustering and consequently does not suffer from the granularity issues.

We would like to stress that the difference between CRM and previously discussed models is not merely conceptual. In section 4 we will show that CRM performs significantly better than all previosly proposed models on the tasks of image annotation and retrieval. To ensure a fair comparison, we use exactly the same data set and same feature representations as were used in [3, 4, 5, 9].

## 3 A Model of Annotated Images

The purpose of this section is to introduce a statistical formalism that will allow us to model a relationship between the contents of a given image and the annotation of that image. We will describe an approach to learning a joint probability disdribution $P(\mathbf{r}, \mathbf{w})$ over the regions $\mathbf{r}$ of some image and the words $\mathbf{w}$ in its annotation. Knowing the joint distribution $P(\mathbf{r}, \mathbf{w})$ is the key to solving two important real-world problems:

1. **Image Annotation.** Suppose we are given a new image for which no annotation is provided. That is, we know $\mathbf{r}$, but do not know $\mathbf{w}$. Having a joint distribution allows us to compute a conditional likelihood $P(\mathbf{w}|\mathbf{r})$ which can then be used to guess the most likely annotation $\mathbf{w}$ for the image in question. The new annotation can be presented to a user, indexed, or used for retrieval purposes.
2. **Image Retrieval.** Suppose we are given a collection of un-annotated images and a text query $\mathbf{w}_{qry}$ consisting of a few keywords. Knowing the joint model of images and annotations, we can compute the query likelihood $P(\mathbf{w}_{qry}|\mathbf{r}_J)$ for every image $J$ in the dataset. We can then rank images in the collection according to their likelihood of having the query as annotation, resulting in a special case of the popular Language Modeling approach to Information Retrieval [6].

The remainder of this section is organized as follows. In section 3.1 we discuss our choice of representation for images and their annotations. Section 3.2 presents a generative framework for relating image regions with image annotations. Section 3.3 provides detailed estimates for the components of our model.

### 3.1 Representation of Images and Annotations

Let $\mathcal{C}$ denote the finite set of all possible pixel colors. We assume that $\mathcal{C}$ includes one "transparent" color $c_0$, which will be handy when we have to layer image regions. As

a matter of convenience, we assume that all images are of a fixed size $W \times H$.[1] This assumption allows us to represent any image as an element of a finite set $\mathcal{R} = \mathcal{C}^{W \times H}$. We assume that each image contains several distinct *regions* $\{r_1 \ldots r_n\}$. Each region is itself an element of $\mathcal{R}$ and contains the pixels of some prominent object in the image, all pixels around the object are set to be transparent. For example, in Figure 1 we have a hypothetical picture containing three prominent objects: a tiger, the sun and some grass. Each object is represented by its own region: $r_1$ for the sun, $r_2$ for the grass, and $r_3$ for the tiger. The final image is the result of stacking or layering the regions on top of each other, as shown on the right side of Figure 1.

In our model of images, a central part will be played by a special function $\mathcal{G}$ which maps image regions $r \in \mathcal{R}$ to real-valued vectors $g \in \mathbb{R}^k$. The value $\mathcal{G}(r)$ represents a set of *features*, or *characteristics* of an image region. The features could reflect the position of an object region, its relative size, a crude reflection of shape, as well as predominant colors and textures. For example, in Figure 1 the region $r_1$ (sun) is a round object, located in the upper-right portion of the image, yellowish in color with a smooth texture. When we model image generation we will treat the output of $\mathcal{G}$ as a *generator* or a "recipe" for producing a certain type of image. For example, a feature vector $g_1 = \mathcal{G}(r_1)$ can be thought of as a generator for any image region resembling a sun-like object in the upper-left corner.

Finally, an annotation for a given image is a set of words $\{w_1 \ldots w_m\}$ drawn from some finite vocabulary $\mathcal{V}$. We assume that the annotation describes the objects represented by regions $\{r_1 \ldots r_n\}$. However, contrary to prior work [4, 3] we do not assume an underlying one-to-one correspondence between the objects in the image annotation and words in the annotation. Instead, we are interested in modeling a joint probability for observing a set of image regions $\{r_1 \ldots r_n\}$ together with the set of annotation words $\{w_1 \ldots w_m\}$.

### 3.2 A Model for Generating Annotated Images

Suppose $\mathcal{T}$ is the training set of annotated images, and let $J$ be an element of $\mathcal{T}$. According to the previous section $J$ is represented as a set of image regions $\mathbf{r}_J = \{r_1 \ldots r_n\}$ along with the corresponding annotation $\mathbf{w}_J = \{w_1 \ldots w_m\}$. We assume that the process that generated $J$ is based on three distinct probability distributions. First, we assume that the words in $\mathbf{w}_J$ are an i.i.d. random sample from some underlying multinomial distribution $P_\mathcal{V}(\cdot|J)$. Second, the regions $\mathbf{r}_J$ are produced from a corresponding set of generator vectors $g_1 \ldots g_n$ according to a process $P_\mathcal{R}(r_i|g_i)$ which is independent of $J$. Finally, the generator vectors $g_1 \ldots g_n$ are themselves an i.i.d. random sample from some underlying multi-variate density function $P_\mathcal{G}(\cdot|J)$.

Now let $\mathbf{r}_A = \{r_1 \ldots r_{n_A}\}$ denote the regions of some image $A$, which is not in the training set $\mathcal{T}$. Similarly, let $\mathbf{w}_B = \{w_1 \ldots w_{n_B}\}$ be some arbitrary sequence of words. We would like to model $P(\mathbf{r}_A, \mathbf{w}_A)$, the joint probability of observing an image defined by $\mathbf{r}_A$ together with annotation words $\mathbf{w}_B$. We hypothesize that the observation $\{\mathbf{r}_A, \mathbf{w}_B\}$ came from the same process that generated one of the images $J^*$ in the training set $\mathcal{T}$. However, we don't know which process that was, and so we compute an expectation over all images $J \in \mathcal{T}$. The overall process for jointly generating $\mathbf{w}_B$ and $\mathbf{r}_A$ is as follows:

1. Pick a training image $J \in \mathcal{T}$ with some probability $P_\mathcal{T}(J)$

2. For $b = 1 \ldots n_B$:

    (a) Pick the annotation word $w_b$ from the multinomial distribution $P_\mathcal{V}(\cdot|J)$.

3. For $a = 1 \ldots n_A$:

    (a) Sample a generator vector $g_a$ from the probability density $P_\mathcal{G}(\cdot|J)$.

    (b) Pick the image region $r_a$ according to the probability $P_\mathcal{R}(r_a|g_a)$

Figure 1 shows a graphical dependency diagram for the generative process outlined above. We show the process of generating a simple image consisting of three regions and a corresponding 3-word annotation. Note that the number of words in the annotation $n_B$ does not have to be the same as the number of image regions $n_A$. Formally, the probability of a joint observation $\{\mathbf{r}_A, \mathbf{w}_B\}$ is given by:

$$P(\mathbf{r}_A, \mathbf{w}_B) = \sum_{J \in \mathcal{T}} P_{\mathcal{T}}(J) \prod_{b=1}^{n_B} P_{\mathcal{V}}(w_b|J) \prod_{a=1}^{n_A} \int_{I\!\!R^k} P_{\mathcal{R}}(r_a|g_a) P_{\mathcal{G}}(g_a|J) dg_a \qquad (1)$$

### 3.3 Estimating Parameters of the Model

In this section we will discuss simple but effective estimation techniques for the four components of the model: $P_{\mathcal{T}}$, $P_{\mathcal{V}}$, $P_{\mathcal{G}}$ and $P_{\mathcal{R}}$. $P_{\mathcal{T}}(J)$ is the probability of selecting the underlying model of image $J$ to generate some new observation $\mathbf{r}, \mathbf{w}$. In the absence of any task knowledge we use a uniform prior $P_{\mathcal{T}}(J) = 1/N_{\mathcal{T}}$, where $N_{\mathcal{T}}$ is the size of the training set.

$P_{\mathcal{R}}(r|g)$ is a global probability distribution responsible for mapping generator vectors $g \in I\!\!R^k$ to actual image regions $r \in \mathcal{R}$. In our case for every image region $r$ there is only one corresponding generator $g = \mathcal{G}(r)$, so we can assume a particularly simple form for the distribution $P_{\mathcal{R}}$:

$$P_{\mathcal{R}}(r|g) = \begin{cases} 1/N_g & \text{if } \mathcal{G}(r) = g \\ 0 & \text{otherwise} \end{cases} \qquad (2)$$

where $N_g$ is the number of all regions $r'$ in $\mathcal{R}$ such that $\mathcal{G}(r') = g$. For the scope of the current paper we do not attempt to reliably estimate $N_g$, instead we assume it to be a constant independent of $g$.

$P_{\mathcal{G}}(\cdot|J)$ is a density function responsible for generating the feature vectors $g_1 \ldots g_n$, which are later mapped to image regions $\mathbf{r}_J$ according to $P_{\mathcal{R}}$. We use a non-parametric kernel-based density estimate for the distribution $P_{\mathcal{G}}$. Assuming $\mathbf{r}_J = \{r_1 \ldots r_n\}$ to be the set of regions of image $J$ we estimate:

$$P_{\mathcal{G}}(g|J) = \frac{1}{n} \sum_{i=1}^{n} \frac{1}{\sqrt{2^k \pi^k |\Sigma|}} \exp \left\{ (g - \mathcal{G}(r_i))^{\top} \Sigma^{-1} (g - \mathcal{G}(r_i)) \right\} \qquad (3)$$

Equation (3) arises out of placing a Gaussian kernel over the feature vector $\mathcal{G}(r_i)$ of every region of image $J$. Each kernel is parametrized by the feature covariance matrix $\Sigma$. As a matter of convenience we assumed $\Sigma = \beta \cdot I$, where $I$ is the identity matrix. $\beta$ playes the role of kernel *bandwidth*: it determines the smoothness of $P_{\mathcal{G}}$ around the support point $\mathcal{G}(r_i)$. The value of $\beta$ is selected empirically on a held-out portion of the training set $\mathcal{T}$.

$P_{\mathcal{V}}(\cdot|J)$ is the multinomial distribution that is assumed to have generated the annotation $\mathbf{w}_J$ of image $J \in \mathcal{T}$. We use a Bayesian framework for estimating $P_{\mathcal{V}}(\cdot|J)$. Let $I\!\!P^{\mathcal{V}}$ be the simplex of all multinomial distributions over $\mathcal{V}$. We assume a Dirichlet prior over $I\!\!P^{\mathcal{V}}$ that has parameters $\{\mu p_v : v \in \mathcal{V}\}$. Here $\mu$ is a constant, selected empirically, and $p_v$ is the relative frequency of observing the word $v$ in the training set. Introducing the observation $\mathbf{w}_J$ results in a Dirichlet posterior over $I\!\!P^{\mathcal{V}}$ with parameters $\{\mu p_v + N_{v,J} : v \in \mathcal{V}\}$. Here $N_{v,J}$ is the number of times $v$ occurs in the observation $\mathbf{w}_J$. Computing the expectation over this Dirichlet posterior gives us the following Bayesian estimate for $P_{\mathcal{V}}$:

$$P_{\mathcal{V}}(v|J) = \frac{\mu p_v + N_{v,J}}{\mu + \sum_{v'} N_{v',J}} \qquad (4)$$

## 4 Experimental Results

To provide a meaningful comparison with previously-reported results, we use, without any modification, the dataset provided by Duygulu *et al.*[4] [2]. This allows us to compare the

| Models | Co-occurence | Translation | CMRM | CRM | |
|---|---|---|---|---|---|
| #words with recall $\geq$ 0 | 19 | 49 | 66 | 107 | +62% |
| Results on 49 best words, as in[1, 5] | | | | | |
| Mean per-word Recall | - | 0.34 | 0.48 | 0.70 | +46% |
| Mean per-word Precision | - | 0.20 | 0.40 | 0.59 | +48% |
| Results on all 260 words | | | | | |
| Mean per-word Recall | 0.02 | 0.04 | 0.09 | 0.19 | +111% |
| Mean per-word Precision | 0.03 | 0.06 | 0.10 | 0.16 | +60 % |

Table 1: Comparing recall and precision of the four models on the task of automatic image annotation. Our model (CRM) substantially outperforms all other models. Percent improvements are over the best previously-reported results (CMRM).

performance of models in a strictly controlled manner. The dataset consists of 5,000 images from 50 Corel Stock Photo cds. Each cd includes 100 images on the same topic. Each image contains an annotation of 1-5 keywords. Overall there are 371 words. Prior to modeling, every image in the dataset is pre-segmented into regions using general-purpose algorithms, such as normalized cuts [11]. We use pre-computed feature vector $\mathcal{G}(r)$ for every segmented region $r$. The feature set consists of 36 features: 18 color features, 12 texture features and 6 shape features. For details of the features refer to [4]. Since we directly model the generation of feature vectors, there is no need to quantize feature data, as was done in [1, 4, 5]. We divided the dataset into 3 parts - with 4,000 training set images, 500 evaluation set images and 500 images in the test set. The evaluation set is used to find system parameters. After fixing the parameters, we merged the 4,000 training set and 500 evaluation set images to make a new training set. This corresponds to the training set of 4500 images and the test set of 500 images used by Duygulu *et al* [4].

### 4.1   Results: Automatic Image Annotation

In this section we evaluate the performance of our model on the task of automatic image annotation. We are given an un-annotated image $J$ and are asked to automatically produce an annotation $\mathbf{w}_{auto}$. The automatic annotation is then compared to the held-out human annotation $\mathbf{w}_J$. We follow the experimental methodology used by[4, 5]. Given a set of image regions $\mathbf{r}_J$ we use equation (1) to arrive at the conditional distribution $P(w|\mathbf{r}_J)$. We take the top 5 words from that distribution and call them the automatic annotation of the image in question. Then, following [4], we compute annotation recall and precision for every word in the testing set. Recall is the number of images correctly annotated with a given word, divided by the number of images that have that word in the human annotation. Precision is the number of correctly annotated images divided by the total number of images annotated with that particular word (correctly or not). Recall and precision values are averaged over the set of testing words.

We compare the annotation performance of the four models: the Co-occurrence Model [9], the Translation Model [4], CMRM [5] and the model proposed in this paper (CRM). We report the results on two sets of words: the subset of *49 best* words which was used by[4, 5], and the complete set of all 260 words that occur in the testing set. Table 1 shows the performance on both word sets. The figures clearly show that the model presented here (CRM) substabtially outperforms the other models and is the only one of the four capable of producing reasonable mean recall and mean precision numbers when every word in the test set is used. In Figure2 we provide sample annotations for the two best models in the table, CMRM and CRM, showing that the model in this paper is considerably more accurate.

| Images |  |  |  |  |
|---|---|---|---|---|
| CMRM Annotation | water sky plane bear | water sky plane jet tree | water sky tree people | people rocks water buildings |
| CRM Annotation | lizard marine iguana rocks | snow bear polar tundra | train railroad tracks locomotive | cat tiger water forest |

Figure 2: The generative model based on contiuous features (CRM) that is proposed here performs substantially better than the discrete cross-media relevance model (CMRM) for annotating images in the test set.

| Query length | 1 word | 2 words | 3 words | 4 words |
|---|---|---|---|---|
| Number of queries | 179 | 386 | 178 | 24 |
| Relevant images | 1675 | 1647 | 542 | 67 |
| Precision after 5 retrieved images | | | | |
| CMRM | 0.1989 | 0.1306 | 0.1494 | 0.2083 |
| CRM | 0.2480 **+25%** | 0.1902 **+45%** | 0.1888 **+26%** | 0.2333 +12% |
| Mean Average Precision | | | | |
| CMRM | 0.1697 | 0.1642 | 0.2030 | 0.2765 |
| CRM | 0.2353 **+39%** | 0.2534 **+54%** | 0.3152 **+55%** | 0.4471 +61% |

Table 2: Comparing our model to the Cross-Media Relevance Model (CMRM) on the task of image retrieval. Our model outperforms the CMRM model by a wide margin on all query sets. Boldface figures mark improvements that are statistically significant according to sign test with a confidence of 99% ($p$-value $< 0.01$).

## 4.2 Results: Ranked Retrieval of Images

In this section we turn our attention to the problem of ranked retrieval of images. In the retrieval setting we are given a text query $\mathbf{w}_{qry}$ and a testing collection of un-annotated images. For each testing image $J$ we use equation (1) to get the conditional probability $P(\mathbf{w}_{qry}|\mathbf{r}_J)$. All images in the collection are ranked according to the conditional likelihood $P(\mathbf{w}_{qry}|\mathbf{r}_J)$. This can be thought of as a special case of the popular Langauge Modeling approach to Information Retrieval, proposed by Ponte and Croft[6]. In our retrieval experiments we do our best to reproduce the same settings that were used by Jeon et.al[5] in their work. Following[5], we use four sets of queries, constructed from all 1-, 2-, 3- and 4-word combinations of words that occur at least twice in the testing set. An image is considered relevant to a given query if its *manual* annotation contains all of the query words. As our evaluation metrics we use precision at 5 retrieved images and non-interpolated average precision[3], averaged over the entire query set. Precision at 5 documents is a good measure of performance for a casual user who is interested in retrieving a couple of relevant items without looking at too much junk. Average precision is more appropriate for a professional user who wants to find a large proportion of relevant items.

Table 2 shows the performance of our model on the four query sets, contrasted with performance of the CMRM[5] baseline on the same data. Baseline performance figures are quoted directly from the tables in[5]. We observe that our model substantially outperforms the CMRM baseline on every query set. Improvements in average precision are particularly impressive, our model outperforms the baseline by 40 - 60 percent. All improvements on 1-, 2- and 3-word queries are statistically significant based on a sign test with a $p$-value of

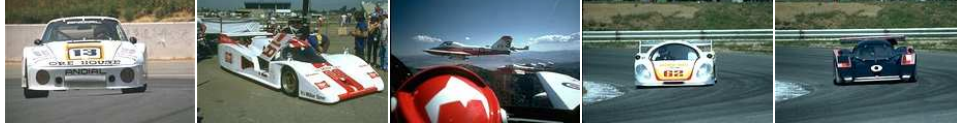

Figure 3: Example: top 5 images retrieved in responce to text query "cars track"

0.01. We are also very encouraged by the precision our model shows at 5 retrieved images: precision values around 0.2 suggest that an average query always has a relevant image in the top 5. Figure 3 shows top 5 images retrieved in response to the text query "cars track".

## 5  Conclusions and Future Work

We have proposed a new statistical generative model for learning the semantics of images. We showed that this model works significantly better than a number of other models for image annotation and retrieval. Our model works directly on the continuous features. Future work will include the extension of this work to larger datasets (both training and test data). We believe this is needed both for better coverage and an evaluation of how such algorithms extend to large data sets. Improved feature sets may also lead to substantial improvements in performance.

## 6  Acknowledgments

We thank Kobus Barnard for making their dataset [4] available. This work was supported in part by the Center for Intelligent Information Retrieval, by the National Science Foundation under grant NSF IIS-9909073 and by SPAWARSYSCEN-SD under grants N66001-99-1-8912 and N66001-02-1-8903. Jiwoon Jeon is partially supported by the Government of Korea. Any opinions, findings and conclusions or recommendations expressed in this material are the author(s) and do not necessarily reflect those of the sponsor.

## Footnotes

[1] The assumptions of finite colormap and fixed image size can easily be relaxed but require arguments that are beyond the scope of this paper.

[2] Available at http://www.cs.arizona.edu/people/kobus/ research/data/eccv_2002

[3]Average precision is the average of precision values at the ranks where relevant items occur.

## References

[1] K. Barnard, P. Duygulu, N. de Freitas, D. Forsyth, D. Blei, and M. I. Jordan. Matching words and pictures. *Journal of Machine Learning Research*, 3:1107-1135, 2003.

[2] D. Blei (2003) Private Communication.

[3] D. Blei, and M. I. Jordan. (2003) Modeling annotated data. In *Proceedings of the 26th Intl. ACM SIGIR Conf.*, pages 127–134, 2003

[4] P. Duygulu, K. Barnard, N. de Freitas, and D. Forsyth. Object recognition as machine translation: Learning a lexicon for a fixed image vocabulary. In *Seventh European Conf. on Computer Vision*, pages 97-112, 2002.

[5] J. Jeon, V. Lavrenko and R. Manmatha. (2003) Automatic Image Annotation and Retrieval using Cross-Media Relevance Models In *Proceedings of the 26th Intl. ACM SIGIR Conf.*, pages 119–126, 2003

[6] Ponte, J. M. and Croft, W. B. (1998). A language modeling approach to information retrieval. Proceedings of the 21st Intl. ACM SIGIR Conf., pages 275–281.

[7] V. Lavrenko and W. Croft. Relevance-based language models. *Proceedings of the 24th Intl. ACM SIGIR Conf.*, pages 120-127, 2001.

[8] V. Lavrenko, M. Choquette, and W. Croft. Cross-lingual relevance models. *Proceedings of the 25th Intl. ACM SIGIR Conf.*, pages 175–182, 2002.

[9] Y. Mori, H. Takahashi, and R. Oka. Image-to-word transformation based on dividing and vector quantizing images with words. In *MISRM'99 First Intl. Workshop on Multimedia Intelligent Storage and Retrieval Management*, 1999.

[10] H. Schneiderman, T. Kanade. A Statistical Method for 3D Object Detection Applied to Faces and Cars. Proc. IEEE CVPR 2000: 1746-1759

[11] J. Shi and J. Malik. Normalized cuts and image segmentation. *IEEE Transactions on Pattern Analysis and Machine Intelligence*, 22(8):888–905, 2000.
